# Shadow Dirichlet for Restricted Probability Modeling

**Bela A. Frigyik, Maya R. Gupta, and Yihua Chen**
Department of Electrical Engineering
University of Washington
Seattle, WA 98195
`frigyik@gmail.com, gupta@ee.washington.edu, yihuachn@gmail.com`

## Abstract

Although the Dirichlet distribution is widely used, the independence structure of its components limits its accuracy as a model. The proposed shadow Dirichlet distribution manipulates the support in order to model probability mass functions (pmfs) with dependencies or constraints that often arise in real world problems, such as regularized pmfs, monotonic pmfs, and pmfs with bounded variation. We describe some properties of this new class of distributions, provide maximum entropy constructions, give an expectation-maximization method for estimating the mean parameter, and illustrate with real data.

## 1 Modeling Probabilities for Machine Learning

Modeling probability mass functions (pmfs) as random is useful in solving many real-world problems. A common random model for pmfs is the Dirichlet distribution [1]. The Dirichlet is conjugate to the multinomial and hence mathematically convenient for Bayesian inference, and the number of parameters is conveniently linear in the size of the sample space. However, the Dirichlet is a distribution over the entire probability simplex, and for many problems this is simply the wrong domain if there is application-specific prior knowledge that the pmfs come from a restricted subset of the simplex.

For example, in natural language modeling, it is common to regularize a pmf over n-grams by some generic language model distribution $q_0$, that is, the pmf to be modeled is assumed to have the form $\theta = \lambda q + (1 - \lambda)q_0$ for some $q$ in the simplex, $\lambda \in (0, 1)$ and a fixed generic model $q_0$ [2]. But once $q_0$ and $\lambda$ are fixed, the pmf $\theta$ can only come from a subset of the simplex. Another natural language processing example is modeling the probability of keywords in a dictionary where some words are related, such as `espresso` and `latte`, and evidence for the one is to some extent evidence for the other. This relationship can be captured with a bounded variation model that would constrain the modeled probability of `espresso` to be within some $\epsilon$ of the modeled probability of `latte`. We show that such bounds on the variation between pmf components also restrict the domain of the pmf to a subset of the simplex. As a third example of restricting the domain, the similarity discriminant analysis classifier estimates class-conditional pmfs that are constrained to be monotonically increasing over an ordered sample space of discrete similarity values [3].

In this paper we propose a simple variant of the Dirichlet whose support is a subset of the simplex, explore its properties, and show how to learn the model from data. We first discuss the alternative solution of renormalizing the Dirichlet over the desired subset of the simplex, and other related work. Then we propose the *shadow Dirichlet* distribution; explain how to construct a shadow Dirichlet for three types of restricted domains: the regularized pmf case, bounded variation between pmf components, and monotonic pmfs; and discuss the most general case. We show how to use the expectation-maximization (EM) algorithm to estimate the shadow Dirichlet parameter $\alpha$, and present simulation results for the estimation.

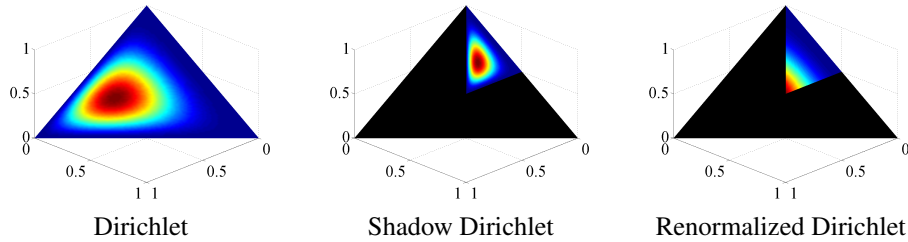

Figure 1: Dirichlet, shadow Dirichlet, and renormalized Dirichlet for $\alpha = [3.94\ 2.25\ 2.81]$.

## 2 Related Work

One solution to modeling pmfs on only a subset of the simplex is to simply restrict the support of the Dirichlet to the desired support $\tilde{\mathcal{S}}$, and renormalize the Dirichlet over $\tilde{\mathcal{S}}$ (see Fig. 1 for an example). This renormalized Dirichlet has the advantage that it is still a conjugate distribution for the multinomial. Nallapati et al.considered the renormalized Dirichlet for language modeling, but found it difficult to use because the density requires numerical integration to compute the normalizer [4] . In addition, there is no closed form solution for the mean, covariance, or peak of the renormalized Dirichlet, making it difficult to work with. Table 1 summarizes these properties. Additionally, generating samples from the renormalized Dirichlet is inefficient: one draws samples from the standard Dirichlet, then rejects realizations that are outside $\tilde{\mathcal{S}}$. For high-dimensional sample spaces, this could greatly increase the time to generate samples.

Although the Dirichlet is a classic and popular distribution on the simplex, Aitchison warns it "is totally inadequate for the description of the variability of compositional data," because of its "implied independence structure and so the Dirichlet class is unlikely to be of any great use for describing compositions whose components have even weak forms of dependence" [5]. Aitchison instead championed a logistic normal distribution with more parameters to control covariance between components.

A number of variants of the Dirichlet that can capture more dependence have been proposed and analyzed. For example, the scaled Dirichlet enables a more flexible shape for the distribution [5], but does not change the support. The original Dirichlet$(\alpha_1, \alpha_2, \ldots \alpha_d)$ can be derived as $Y_j / \sum_j Y_j$ where $Y_j \sim \Gamma(\alpha_j, \beta)$, whereas the scaled Dirichlet is derived from $Y_j \sim \Gamma(\alpha_j, \beta_j)$, resulting in density $p(\theta) = \gamma \prod_j \frac{\beta_j^{\alpha_j} \theta_j^{\alpha_j - 1}}{(\sum_i \beta_i \theta_i)^{\alpha_1 + \cdots + \alpha_d}}$, where $\beta, \alpha \in \mathbb{R}^d_+$ are parameters, and $\gamma$ is the normalizer.

Another variant is the generalized Dirichlet [6] which also has parameters $\beta, \alpha \in \mathbb{R}^d_+$, and allows greater control of the covariance structure, again without changing the support. As perhaps first noted by Karl Pearson [7] and expounded upon by Aitchison [5], correlations of proportional data can be very misleading. Many Dirichlet variants have been generalizations of the Connor-Mossiman variant, Dirichlet process variants, other compound Dirichlet models, and hierarchical Dirichlet models. Ongaro et al. [8] propose the *flexible Dirichlet distribution* by forming a re-parameterized mixture of Dirichlet distributions. Rayens and Srinivasan [9] considered the dependence structure for the general Dirichlet family called the generalized Liouville distributions. In contrast to prior efforts, the shadow Dirichlet manipulates the support to achieve various kinds of dependence that arise frequently in machine learning problems.

## 3 Shadow Dirichlet Distribution

We introduce a new distribution that we call the *shadow Dirichlet* distribution. Let $\mathcal{S}$ be the probability $(d-1)$-simplex, and let $\tilde{\Theta} \in \mathcal{S}$ be a random pmf drawn from a Dirichlet distribution with density $p_D$ and unnormalized parameter $\alpha \in \mathbb{R}^d_+$. Then we say the random pmf $\Theta \in \mathcal{S}$ is distributed according to a shadow Dirichlet distribution if $\Theta = M\tilde{\Theta}$ for some fixed $d \times d$ left-stochastic (that is, each column of $M$ sums to 1) full-rank (and hence invertible) matrix $M$, and we call $\tilde{\Theta}$ the *gen-*

*erating Dirichlet* of $\Theta$, or $\Theta$'s *Dirichlet shadow*. Because $M$ is a left-stochastic linear map between finite-dimensional spaces, it is a continuous map from the convex and compact $\mathcal{S}$ to a convex and compact subset of $\mathcal{S}$ that we denote $\mathcal{S}_M$.

The shadow Dirichlet has two parameters: the generating Dirichlet's parameter $\alpha \in \mathbb{R}_+^d$, and the $d \times d$ matrix $M$. Both $\alpha$ and $M$ can be estimated from data. However, as we show in the following subsections, the matrix $M$ can be profitably used as a design parameter that is chosen based on application-specific knowledge or side-information to specify the restricted domain $\mathcal{S}_M$, and in that way impose dependency between the components of the random pmfs.

The shadow Dirichlet density $p(\theta)$ is the normalized pushforward of the Dirichlet density, that is, it is the composition of the Dirichlet density and $M^{-1}$ with the Jacobian:

$$p(\theta) = \frac{1}{B(\alpha)\,|\det(M)|} \prod_j (M^{-1}\theta)_j^{\alpha_j - 1}, \tag{1}$$

where $B(\alpha) \triangleq \frac{\prod_j \Gamma(\alpha_j)}{\Gamma(\alpha_0)}$ is the standard Dirichlet normalizer, and $\alpha_0 = \sum_{j=1}^d \alpha_j$ is the standard Dirichlet precision factor. Table 1 summarizes the basic properties of the shadow Dirichlet. Fig. 1 shows an example shadow Dirichlet distribution.

Generating samples from the shadow Dirichlet is trivial: generate samples from its generating Dirichlet (for example, using stick-breaking or urn-drawing) and multiply each sample by $M$ to create the corresponding shadow Dirichlet sample.

Table 1: Table compares and summarizes the Dirichlet, renormalized Dirichlet, and shadow Dirichlet distributions.

| | Dirichlet($\alpha$) | Shadow Dirichlet ($\alpha$, $M$) | Renormalized Dirichlet ($\alpha$, $\tilde{\mathcal{S}}$) |
|---|---|---|---|
| **Density** $p(\theta)$ | $\frac{1}{B(\alpha)} \prod_{j=1}^d \theta_j^{\alpha_j-1}$ | $\frac{1}{B(\alpha)|\det(M)|} \prod_{j=1}^d (M^{-1}\theta)_j^{\alpha_j-1}$ | $\frac{1}{\int_{\tilde{\mathcal{S}}} \prod_{j=1}^d q_j^{\alpha_j-1} dq} \prod_{j=1}^d \theta_j^{\alpha_j-1}$ |
| **Mean** | $\frac{\alpha}{\alpha_0}$ | $M \frac{\alpha}{\alpha_0}$ | $\int_{\tilde{\mathcal{S}}} \theta p(\theta) d\theta$ |
| **Covariance** | $\mathrm{Cov}(\Theta)$ | $M\,\mathrm{Cov}(\Theta) M^T$ | $\int_{\tilde{\mathcal{S}}} (\theta - \bar{\theta})(\theta - \bar{\theta})^T p(\theta) d\theta$ |
| **Mode** (if $\alpha > 1$) | $\frac{\alpha_j-1}{\alpha_0-d}$ | $M \frac{\alpha_j-1}{\alpha_0-d}$ | $\max_{\theta \in \tilde{\mathcal{S}}} p(\theta)$ |
| **How to Sample** | stick-breaking, urn-drawing | draw from Dirichlet($\alpha$), multiply by $M$ | draw from Dirichlet($\alpha$), reject if not in $\tilde{\mathcal{S}}$ |
| **ML Estimate** | iterative (simple functions) | iterative (simple functions) | unknown complexity |
| **ML Compound Estimate** | iterative (simple functions) | iterative (numerical integration) | unknown complexity |

### 3.1 Example: Regularized Pmfs

The shadow Dirichlet can be designed to specify a distribution over a set of regularized pmfs $\mathcal{S}_M = \{\theta \mid \theta = \lambda\tilde{\theta} + (1-\lambda)\breve{\theta}, \tilde{\theta} \in \mathcal{S}\}$, for specific values of $\lambda$ and $\breve{\theta}$. In general, for a given $\lambda$ and $\breve{\theta} \in \mathcal{S}$, the following $d \times d$ matrix $M$ will change the support to the desired subset $\mathcal{S}_M$ by mapping the extreme points of $\mathcal{S}$ to the extreme points of $\mathcal{S}_M$:

$$M = (1-\lambda)\breve{\theta}\mathbf{1}^T + \lambda I, \tag{2}$$

where $I$ is the $d \times d$ identity matrix. In Section 4 we show that the $M$ given in (2) is optimal in a maximum entropy sense.

## 3.2 Example: Bounded Variation Pmfs

We describe how to use the shadow Dirichlet to model a random pmf that has bounded variation such that $|\theta_k - \theta_l| \leq \epsilon_{k,l}$ for any $k, \ell \in \{1, 2, \ldots, d\}$ and $\epsilon_{k,l} > 0$. To construct specified bounds on the variation, we first analyze the variation for a given $M$. For any $d \times d$ left stochastic matrix $M$, $\theta = M\tilde{\theta} = \left[ \sum_{j=1}^{d} M_{1j}\tilde{\theta}_j \quad \ldots \quad \sum_{j=1}^{d} M_{dj}\tilde{\theta}_j \right]^T$, so the difference between any two entries is

$$|\theta_k - \theta_l| = \left| \sum_j (M_{kj} - M_{lj})\tilde{\theta}_j \right| \leq \sum_j |M_{kj} - M_{lj}| \, \tilde{\theta}_j. \tag{3}$$

Thus, to obtain a distribution over pmfs with bounded $|\theta_k - \theta_\ell| \leq \epsilon_{k,l}$ for any $k, \ell$ components, it is sufficient to choose components of the matrix $M$ such that $|M_{kj} - M_{lj}| \leq \epsilon_{k,l}$ for all $j = 1, \ldots, d$ because $\tilde{\theta}$ in (3) sums to 1.

One way to create such an $M$ is using the regularization strategy described in Section 3.1. For this case, the $j$th component of $\theta$ is $\theta_j = \left( M\tilde{\theta} \right)_j = \lambda\tilde{\theta}_j + (1 - \lambda)\breve{\theta}_j$, and thus the variation between the $i$th and $j$th component of any pmf in $\mathcal{S}_M$ is:

$$|\theta_i - \theta_j| = \left| \lambda\tilde{\theta}_i + (1 - \lambda)\breve{\theta}_i - \lambda\tilde{\theta}_j - (1 - \lambda)\breve{\theta}_j \right| \leq \lambda \left| \tilde{\theta}_i - \tilde{\theta}_j \right| + (1 - \lambda) \left| \breve{\theta}_i - \breve{\theta}_j \right|$$

$$\leq \lambda + (1 - \lambda) \max_{i,j} \left| \breve{\theta}_i - \breve{\theta}_j \right|. \tag{4}$$

Thus by choosing an appropriate $\lambda$ and regularizing pmf $\breve{\theta}$, one can impose the bounded variation given by (4). For example, set $\breve{\theta}$ to be the uniform pmf, and choose any $\lambda \in (0, 1)$, then the matrix $M$ given by (2) will guarantee that the difference between any two entries of any pmf drawn from the shadow Dirichlet $(M, \alpha)$ will be less than or equal to $\lambda$.

## 3.3 Example: Monotonic Pmfs

For pmfs over ordered components, it may be desirable to restrict the support of the random pmf distribution to only monotonically increasing pmfs (or to only monotonically decreasing pmfs).

A $d \times d$ left-stochastic matrix $M$ that will result in a shadow Dirichlet that generates only monotonically increasing $d \times 1$ pmfs has $k$th column $[0 \ldots 0 \; 1/(d - k + 1) \ldots 1/(d - k + 1)]^T$, we call this the *monotonic M*. It is easy to see that with this $M$ only monotonic $\theta$'s can be produced, because $\theta_1 = \frac{1}{d}\tilde{\theta}_1$ which is less than or equal to $\theta_2 = \frac{1}{d}\tilde{\theta}_1 + \frac{1}{d-1}\tilde{\theta}_2$ and so on. In Section 4 we show that the monotonic $M$ is optimal in a maximum entropy sense.

Note that to provide support over both monotonically increasing and decreasing pmfs with one distribution is not achievable with a shadow Dirichlet, but could be achieved by a mixture of two shadow Dirichlets.

## 3.4 What Restricted Subsets are Possible?

Above we have described solutions to construct $M$ for three kinds of dependence that arise in machine learning applications. Here we consider the more general question: *What subsets of the simplex can be the support of the shadow Dirichlet, and how to design a shadow Dirichlet for a particular support?* For any matrix $M$, by the Krein-Milman theorem [10], $\mathcal{S}_M = M\mathcal{S}$ is the convex hull of its extreme points. If $M$ is injective, the extreme points of $\mathcal{S}_M$ are easy to specify, as a $d \times d$ matrix $M$ will have $d$ extreme points that occur for the $d$ choices of $\theta$ that have only one nonzero component, as the rest of the $\theta$ will create a non-trivial convex combination of the columns of $M$, and therefore cannot result in extreme points of $\mathcal{S}_M$ by definition. That is, the extreme points of $\mathcal{S}_M$ are the $d$ columns of $M$, and one can design any $\mathcal{S}_M$ with $d$ extreme points by setting the columns of $M$ to be those extreme pmfs.

However, if one wants the new support to be a polytope in the probability $(d - 1)$-simplex with $m > d$ extreme points, then one must use a fat $M$ with $d \times m$ entries. Let $\mathcal{S}^m$ denote the probability

$(m-1)$-simplex, then the domain of the shadow Dirichlet will be $M\mathcal{S}^m$, which is the convex hull of the $m$ columns of $M$ and forms a convex polytope in $\mathcal{S}$ with at most $m$ vertices. In this case $M$ cannot be injective, and hence it is not bijective between $\mathcal{S}^m$ and $M\mathcal{S}^m$. However, a density on $M\mathcal{S}^m$ can be defined as:

$$p(\theta) = \frac{1}{B(\alpha)} \int_{\{\tilde{\theta} \mid M\tilde{\theta}=\theta\}} \prod_j \tilde{\theta}_j^{\alpha_j-1} d\tilde{\theta}. \tag{5}$$

On the other hand, if one wants the support to be a low-dimensional polytope subset of a higher-dimensional probability simplex, then a thin $d \times m$ matrix $M$, where $m < d$, can be used to implement this. If $M$ is injective, then it has a left inverse $M^*$ that is a matrix of dimension $m \times d$, and the normalized pushforward of the original density can be used as a density on the image $M\mathcal{S}^m$:

$$p(\theta) = \frac{1}{B(\alpha) \left|\det(M^T M)\right|^{1/2}} \prod_j (M^*\theta)_j^{\alpha_j-1},$$

If $M$ is not injective then one way to determine a density is to use (5).

# 4 Information-theoretic Properties

In this section we note two information-theoretic properties of the shadow Dirichlet. Let $\Theta$ be drawn from shadow Dirichlet density $p_M$, and let its generating Dirichlet $\tilde{\Theta}$ be drawn from $p_D$. Then the differential entropy of the shadow Dirichlet is $h(p_M) = \log|\det(M)| + h(p_D)$, where $h(p_D)$ is the differential entropy of its generating Dirichlet. In fact, the shadow Dirichlet always has less entropy than its Dirichlet shadow because $\log|\det(M)| \leq 0$, which can be shown as a corollary to the following lemma (proof not included due to lack of space):

**Lemma 4.1.** *Let $\{x_1, \ldots, x_n\}$ and $\{y_1, \ldots, y_n\}$ be column vectors in $\mathbb{R}^n$. If each $y_j$ is a convex combination of the $x_i$'s, i.e. $y_j = \sum_{i=1}^n \gamma_{ji} x_i$, $\sum_{i=1}^n \gamma_{ji} = 1$, $\gamma_{jk} \geq 0$, $\forall j, k \in \{1, \ldots, n\}$ then $|\det[y_1, \ldots, y_n]| \leq |\det[x_1, \ldots, x_n]|$.*

It follows from Lemma 4.1 that the constructive solutions for $M$ given in (2) and the monotonic $M$ are optimal in the sense of maximizing entropy:

**Corollary 4.1.** *Let $\mathcal{M}_{reg}$ be the set of left-stochastic matrices $M$ that parameterize shadow Dirichlet distributions with support in $\{\theta \mid \theta = \lambda\tilde{\theta} + (1-\lambda)\breve{\theta}, \tilde{\theta} \in \mathcal{S}\}$, for a specific choice of $\lambda$ and $\breve{\theta}$. Then the $M$ given in (2) results in the shadow Dirichlet with maximum entropy, that is, (2) solves $\arg\max_{M \in \mathcal{M}_{reg}} h(p_M)$.*

**Corollary 4.2.** *Let $\mathcal{M}_{mono}$ be the set of left-stochastic matrices $M$ that parameterize shadow Dirichlet distributions that generate only monotonic pmfs. Then the monotonic $M$ given in Section 3.3 results in the shadow Dirichlet with maximum entropy, that is, the monotonic $M$ solves $\arg\max_{M \in \mathcal{M}_{mono}} h(p_M)$.*

# 5 Estimating the Distribution from Data

In this section, we discuss the estimation of $\alpha$ for the shadow Dirichlet and compound shadow Dirichlet, and the estimation of $M$.

## 5.1 Estimating $\alpha$ for the Shadow Dirichlet

Let matrix $M$ be specified (for example, as described in the subsections of Section 3), and let $q$ be a $d \times N$ matrix where the $i$th column $q_i$ is the $i$th sample pmf for $i = 1 \ldots N$, and let $(q_i)_j$ be the $j$th component of the $i$th sample pmf for $j = 1, \ldots, d$. Then finding the maximum likelihood estimate

of $\alpha$ for the shadow Dirichlet is straightforward:

$$\arg\max_{\alpha\in\mathbb{R}_+^k}\log\prod_{i=1}^{N}p(q_i|\alpha) \equiv \arg\max_{\alpha\in\mathbb{R}_+^k}\log\left[\frac{1}{B(\alpha)\,|\det(M)|}\right]^N + \log\left(\prod_i\prod_j(M^{-1}q_i)_j^{\alpha_j-1}\right)$$

$$\equiv \arg\max_{\alpha\in\mathbb{R}_+^k}\log\left(\frac{1}{B(\alpha)^N}\prod_i\prod_j(\tilde{q}_i)_j^{\alpha_j-1}\right), \tag{6}$$

where $\tilde{q} = M^{-1}q$. Note (6) is the maximum likelihood estimation problem for the Dirichlet distribution given the matrix $\tilde{q}$, and can be solved using the standard methods for that problem (see e.g. [11, 12]).

## 5.2 Estimating $\alpha$ for the Compound Shadow Dirichlet

For many machine learning applications the given data are modeled as samples from realizations of a random pmf, and given these samples one must estimate the random pmf model's parameters. We refer to this case as the compound shadow Dirichlet, analogous to the compound Dirichlet (also called the multivariate Pólya distribution). Assuming one has already specified $M$, we first discuss method of moments estimation, and then describe an expectation-maximization (EM) method for computing the maximum likelihood estimate $\breve{\alpha}$.

One can form an estimate of $\alpha$ by the method of moments. For the standard compound Dirichlet, one treats the samples of the realizations as normalized empirical histograms, sets the normalized $\alpha$ parameter equal to the empirical mean of the normalized histograms, and uses the empirical variances to determine the precision $\alpha_0$. By definition, this estimate will be less likely than the maximum likelihood estimate, but may be a practical short-cut in some cases. For the compound shadow Dirichlet, we believe the method of moments estimator will be a poorer estimate in general. The problem is that if one draws samples from a pmf $\theta$ from a restricted subset $\mathcal{S}_M$ of the simplex, then the normalized empirical histogram $\breve{\theta}$ of those samples may not be in $\mathcal{S}_M$. For example given a monotonic pmf, the histogram of five samples drawn from it may not be monotonic. Then the empirical mean of such normalized empirical histograms may not be in $\mathcal{S}_M$, and so setting the shadow Dirichlet mean $M\alpha$ equal to the empirical mean may lead to an infeasible estimate (one that is outside $\mathcal{S}_M$). A heuristic solution is to project the empirical mean into $\mathcal{S}_M$ first, for example, by finding the nearest pmf in $\mathcal{S}_M$ in squared error or relative entropy. As with the compound Dirichlet, this may still be a useful approach in practice for some problems.

Next we state an EM method to find the maximum likelihood estimate $\breve{\alpha}$. Let $s$ be a $d\times N$ matrix of sample histograms from different experiments, such that the $i$th column $s_i$ is the $i$th histogram for $i=1,\ldots,N$, and $(s_i)_j$ is the number of times we have observed the $j$th event from the $i$th pmf $v_i$. Then the maximum log-likelihood estimate of $\alpha$ solves $\arg\max\log p(s|\alpha)$ for $\alpha\in\mathbb{R}_+^k$. If the random pmfs are drawn from a Dirichlet distribution, then finding this maximum likelihood estimate requires an iterative procedure, and can be done in several ways including a gradient descent (ascent) approach. However, if the random pmfs are drawn from a shadow Dirichlet distribution, then a direct gradient descent approach is highly inconvenient as it requires taking derivatives of numerical integrals. However, it is practical to apply the expectation-maximization (EM) algorithm [13][14], as we describe in the rest of this section. Code to perform the EM estimation of $\alpha$ can be downloaded from idl.ee.washington.edu/publications.php.

We assume that the experiments are independent and therefore $p(s|\alpha) = p(\{s_i\}|\alpha) = \prod_i p(s_i|\alpha)$ and hence $\arg\max_{\alpha\in\mathbb{R}_+^k}\log p(s|\alpha) = \arg\max_{\alpha\in\mathbb{R}_+^k}\sum_i\log p(s_i|\alpha)$.

To apply the EM method, we consider the complete data to be the sample histograms $s$ and the pmfs that generated them $(s, v_1, v_2, \ldots, v_N)$, whose expected log-likelihood will be maximized. Specifically, because of the assumed independence of the $\{v_i\}$, the EM method requires one to repeatedly maximize the *Q-function* such that the estimate of $\alpha$ at the $(m+1)$th iteration is:

$$\alpha^{(m+1)} = \arg\max_{\alpha\in\mathbb{R}_+^k}\sum_{i=1}^{N}E_{v_i|s_i,\alpha^{(m)}}\left[\log p(v_i|\alpha)\right]. \tag{7}$$

Like the compound Dirichlet likelihood, the compound shadow Dirichlet likelihood is not necessarily concave. However, note that the Q-function given in (7) is concave, because $\log p(v_i|\alpha) = -\log|\det(M)| + \log p_{D,\alpha}(M^{-1}v_i)$, where $p_{D,\alpha}$ is the Dirichlet distribution with parameter $\alpha$, and by a theorem of Ronning [11], $\log p_{D,\alpha}$ is a concave function, and adding a constant does not change the concavity. The Q-function is a finite integration of such concave functions and hence also concave [15].

We simplify (7) without destroying the concavity to yield the equivalent problem $\alpha^{(m+1)} = \arg\max g(\alpha)$ for $\alpha \in \mathbb{R}_+^k$, where $g(\alpha) = \log\Gamma(\alpha_0) - \sum_{j=1}^d \log\Gamma(\alpha_j) + \sum_{j=1}^d \beta_j\alpha_j$, and $\beta_j = \frac{1}{N}\sum_{i=1}^N \frac{t_{ij}}{z_i}$, where $t_{ij}$ and $z_i$ are integrals we compute with Monte Carlo integration:

$$t_{ij} = \int_{\mathcal{S}_M} \log(M^{-1}v_i)_j \gamma_i \prod_{k=1}^d (v_i)_k^{(s_i)_k} p_M(v_i|\alpha^{(m)})dv_i$$

$$z_i = \int_{\mathcal{S}_M} \gamma_i \prod_{k=1}^d (v_i)_j k(s_i)_k p_M(v_i|\alpha^{(m)})dv_i,$$

where $\gamma_i$ is the normalization constant for the multinomial with histogram $s_i$.

We apply the Newton method [16] to maximize $g(\alpha)$, where the gradient $\nabla g(\alpha)$ has $k$th component $\psi_0(\alpha_0) - \psi_0(\alpha_1) + \beta_1$, where $\psi_0$ denotes the digamma function. Let $\psi_1$ denote the trigamma function, then the Hessian matrix of $g(\alpha)$ is: $H = \psi_1(\alpha_0)\mathbf{1}\mathbf{1}^T - \operatorname{diag}(\psi_1(\alpha_1),\ldots,\psi_1(\alpha_d))$.

Note that because $H$ has a very simple structure, the inversion of $H$ required by the Newton step is greatly simplified by using the Woodbury identity [17]: $H^{-1} = -\operatorname{diag}(\xi_1,\ldots,\xi_d) - \frac{1}{\xi_0 - \sum_{j=1}^d \xi_j}[\xi_i\xi_j]_{d\times d}$, where $\xi_0 = \frac{1}{\psi_1(\alpha_0)}$ and $\xi_j = \frac{1}{\psi_1(\alpha_j)}$, $j = 1,\ldots,d$.

## 5.3  Estimating $M$ for the Shadow Dirichlet

Thus far we have discussed how to construct $M$ to achieve certain desired properties and how to interpret a given $M$'s effect on the support. In some cases it may be useful to estimate $M$ directly from data, for example, finding the maximum likelihood $M$. In general, this is a non-convex problem because the set of rank $d - 1$ matrices is not convex. However, we offer two approximations. First, note that as in estimating the support of a uniform distribution, the maximum likelihood $M$ will correspond to a support that is no larger than needed to contain the convex hull of sample pmfs. Second, the mean of the empirical pmfs will be in the support, and thus a heuristic is to set the $k$th column of $M$ (which corresponds to the $k$th vertex of the support) to be a convex combination of the $k$th vertex of the standard probability simplex and the empirical mean pmf. We provide code that finds the $d$ optimal such convex combinations such that a specified percentage of the sample pmfs are within the support, which reduces the non-convex problem of finding the maximum likelihood $d \times d$ matrix $M$ to a $d$-dimensional convex relaxation.

# 6  Demonstrations

It is reasonable to believe that if the shadow Dirichlet better matches the problem's statistics, it will perform better in practice, but an open question is how much better? To motivate the reader to investigate this question further in applications, we provide two small demonstrations.

## 6.1  Verifying the EM Estimation

We used a broad suite of simulations to test and verify the EM estimation. Here we include a simple visual confirmation that the EM estimation works: we drew 100 i.i.d. pmfs from a shadow Dirichlet with monotonic $M$ for $d = 3$ and $\alpha = [3.94\ 2.25\ 2.81]$ (used in [18]). From each of the 100 pmfs, we drew 100 i.i.d. samples. Then we applied the EM algorithm to find the $\alpha$ for both the standard compound Dirichlet, and the compound shadow Dirichlet with the correct $M$. Fig. 2 shows the true distribution and the two estimated distributions.

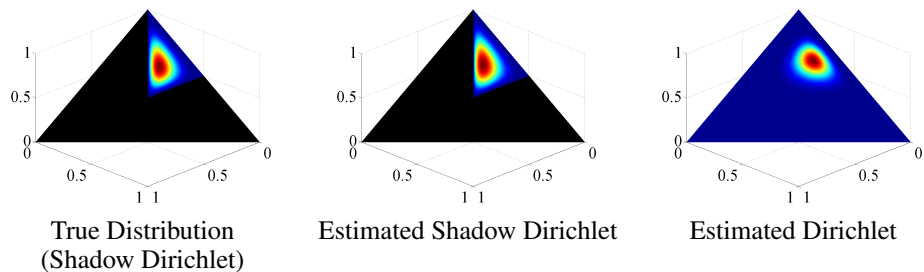

| True Distribution | Estimated Shadow Dirichlet | Estimated Dirichlet |
|---|---|---|
| (Shadow Dirichlet) | | |

Figure 2: Samples were drawn from the true distribution and the given EM method was applied to form the estimated distributions.

## 6.2 Estimating Proportions from Sales

Manufacturers often have constrained manufacturing resources, such as equipment, inventory of raw materials, and employee time, with which to produce multiple products. The manufacturer must decide how to proportionally allocate such constrained resources across their product line based on their estimate of proportional sales. Manufacturer *Artifact Puzzles* gave us their past retail sales data for the 20 puzzles they sold during July 2009 through Dec 2009, which we used to predict the proportion of sales expected for each puzzle. These estimates were then tested on the next five months of sales data, for January 2010 through April 2010. The company also provided a similarity between puzzles $S$, where $S(A, B)$ is the proportion of times an order during the six training months included both puzzle A and B if it included puzzle A. We compared treating each of the six training months of sales data as a sample from a compound Dirichlet versus or a compound shadow Dirichlet. For the shadow Dirichlet, we normalized each column of the similarity matrix $S$ to sum to one so that it was left-stochastic, and used that as the $M$ matrix; this forces puzzles that are often bought together to have closer estimated proportions. We estimated each $\alpha$ parameter by EM to maximize the likelihood of the past sales data, and then estimated the future sales proportions to be the mean of the estimated Dirichlet or shadow Dirichlet distribution. We also compared with treating all six months of sales data as coming from one multinomial which we estimated as the maximum likelihood multinomial, and to taking the mean of the six empirical pmfs.

Table 2: Squared errors between estimates and actual proportional sales.

|      | Multinomial | Mean Pmf | Dirichlet | Shadow Dirichlet |
|------|-------------|----------|-----------|------------------|
| Jan. | .0129       | .0106    | .0109     | **.0093**        |
| Feb. | .0185       | .0206    | .0172     | **.0164**        |
| Mar. | .0231       | .0222    | .0227     | **.0197**        |
| Apr. | .0240       | .0260    | .0235     | **.0222**        |

## 7 Summary

In this paper we have proposed a variant of the Dirichlet distribution that naturally captures some of the dependent structure that arises often in machine learning applications. We have discussed some of its theoretical properties, and shown how to specify the distribution for regularized pmfs, bounded variation pmfs, monotonic pmfs, and for any desired convex polytopal domain. We have derived the EM method and made available code to estimate both the shadow Dirichlet and compound shadow Dirichlet from data. Experimental results demonstrate that the EM method can estimate the shadow Dirichlet effectively, and that the shadow Dirichlet may provide worthwhile advantages in practice.

# References

[1] B. Frigyik, A. Kapila, and M. R. Gupta, "Introduction to the Dirichlet distribution and related processes," Tech. Rep., University of Washington, 2010.

[2] C. Zhai and J. Lafferty, "A study of smoothing methods for language models applied to information retrieval," *ACM Trans. on Information Systems*, vol. 22, no. 2, pp. 179–214, 2004.

[3] Y. Chen, E. K. Garcia, M. R. Gupta, A. Rahimi, and L. Cazzanti, "Similarity-based classification: Concepts and algorithms," *Journal of Machine Learning Research*, vol. 10, pp. 747–776, March 2009.

[4] R. Nallapati, T. Minka, and S. Robertson, "The smoothed-Dirichlet distribution: a building block for generative topic models," Tech. Rep., Microsoft Research, Cambridge, 2007.

[5] Aitchison, *Statistical Analysis of Compositional Data*, Chapman Hall, New York, 1986.

[6] R. J. Connor and J. E. Mosiman, "Concepts of independence for proportions with a generalization of the Dirichlet distibution," *Journal of the American Statistical Association*, vol. 64, pp. 194–206, 1969.

[7] K. Pearson, "Mathematical contributions to the theory of evolution–on a form of spurious correlation which may arise when indices are used in the measurement of organs," *Proc. Royal Society of London*, vol. 60, pp. 489–498, 1897.

[8] A. Ongaro, S. Migliorati, and G. S. Monti, "A new distribution on the simplex containing the Dirichlet family," *Proc. 3rd Compositional Data Analysis Workshop*, 2008.

[9] W. S. Rayens and C. Srinivasan, "Dependence properties of generalized Liouville distributions on the simplex," *Journal of the American Statistical Association*, vol. 89, no. 428, pp. 1465–1470, 1994.

[10] Walter Rudin, *Functional Analysis*, McGraw-Hill, New York, 1991.

[11] G. Ronning, "Maximum likelihood estimation of Dirichlet distributions," *Journal of Statistical Computation and Simulation*, vol. 34, no. 4, pp. 215221, 1989.

[12] T. Minka, "Estimating a Dirichlet distribution," Tech. Rep., Microsoft Research, Cambridge, 2009.

[13] A. P. Dempster, N. M. Laird, and D. B. Rubin, "Maximum likelihood from incomplete data via the EM algorithm," *Journal of the Royal Statistical Society: Series B (Methodological)*, vol. 39, no. 1, pp. 1–38, 1977.

[14] M. R. Gupta and Y. Chen, *Theory and Use of the EM Method*, Foundations and Trends in Signal Processing, Hanover, MA, 2010.

[15] R. T. Rockafellar, *Convex Analysis*, Princeton University Press, Princeton, NJ, 1970.

[16] S. Boyd and L. Vandenberghe, *Convex Optimization*, Cambridge University Press, Cambridge, 2004.

[17] K. B. Petersen and M. S. Pedersen, *Matrix Cookbook*, 2009, Available at matrixcookbook.com.

[18] R. E. Madsen, D. Kauchak, and C. Elkan, "Modeling word burstiness using the Dirichlet distribution," in *Proc. Intl. Conf. Machine Learning*, 2005.

